# Duality, Geometry, and Support Vector Regression

**Jinbo Bi** and **Kristin P. Bennett**
Department of Mathematical Sciences
Rensselaer Polytechnic Institute
Troy, NY 12180
*bij2@rpi.edu, bennek@rpi.edu*

## Abstract

We develop an intuitive geometric framework for support vector regression (SVR). By examining when $\epsilon$-tubes exist, we show that SVR can be regarded as a classification problem in the dual space. Hard and soft $\epsilon$-tubes are constructed by separating the convex or reduced convex hulls respectively of the training data with the response variable shifted up and down by $\epsilon$. A novel SVR model is proposed based on choosing the max-margin plane between the two shifted datasets. Maximizing the margin corresponds to shrinking the effective $\epsilon$-tube. In the proposed approach the effects of the choices of all parameters become clear geometrically.

## 1 Introduction

Support Vector Machines (SVMs) [6] are a very robust methodology for inference with minimal parameter choices. Intuitive geometric formulations exist for the classification case addressing both the error metric and capacity control [1, 2]. For linearly separable classification, the primal SVM finds the separating plane with maximum hard margin between two sets. The equivalent dual SVM computes the closest points in the convex hulls of the data from each class. For the inseparable case, the primal SVM optimizes the soft margin of separation between the two classes. The corresponding dual SVM finds the closest points in the reduced convex hulls. In this paper, we derive analogous arguments for SVM regression (SVR).

We provide a geometric explanation for SVR with the $\epsilon$-insensitive loss function. From the primal perspective, a linear function with no residuals greater than $\epsilon$ corresponds to an $\epsilon$-tube constructed about the data in the space of the data attributes and the response variable [6] (see e.g. Figure 1(a)). The primary contribution of this work is a novel geometric interpretation of SVR from the dual perspective along with a mathematically rigorous derivation of the geometric concepts. In Section 2, for a fixed $\epsilon > 0$ we examine the question "When does a "perfect" or "hard"

$\epsilon$-tube exist?". With duality analysis, the existence of a hard $\epsilon$-tube depends on the separability of two sets. The two sets consist of the training data augmented with the response variable shifted up and down by $\epsilon$. In the dual space, regression becomes the classification problem of distinguishing between these two sets. The geometric formulations developed for the classification case [1] become applicable to the regression case. We call the resulting formulation convex SVR (C-SVR) since it is based on convex hulls of the augmented training data. Much like in SVM classification, to compute a hard $\epsilon$-tube, C-SVR computes the nearest points in the convex hulls of the augmented classes. The corresponding maximum margin (max-margin) planes define the effective $\epsilon$-tube. The size of margin determines how much the effective $\epsilon$-tube shrinks. Similarly, to compute a soft $\epsilon$-tube, reduced-convex SVR (RC-SVR) finds the closest points in the reduced convex hulls of the two augmented sets.

This paper introduces the geometrically intuitive RC-SVR formulation which is a variation of the classic $\epsilon$-SVR [6] and $\nu$-SVR models [5]. If parameters are properly tuned, the methods perform similarly although not necessarily identically. RC-SVR eliminates the pesky parameter $C$ used in $\epsilon$-SVR and $\nu$-SVR. The geometric role or interpretation of $C$ is not known for these formulations. The geometric roles of the two parameters of RC-SVR, $\nu$ and $\epsilon$, are very clear, facilitating model selection, especially for nonexperts. Like $\nu$-SVR, RC-SVR shrinks the $\epsilon$-tube and has a parameter $\nu$ controlling the robustness of the solution. The parameter $\epsilon$ acts as an upper bound on the size of the allowable $\epsilon$-insensitive error function. In addition, RC-SVR can be solved by fast and scalable nearest-point algorithms such as those used in [3] for SVM classification.

## 2   When does a hard $\epsilon$-tube exist?

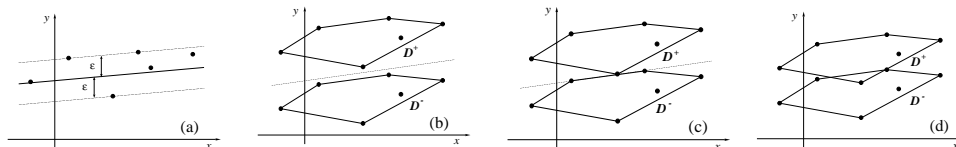

Figure 1: The (a) primal hard $\epsilon_0$-tube, and dual cases: (b) dual strictly separable $\epsilon > \epsilon_0$, (c) dual separable $\epsilon = \epsilon_0$, and (d) dual inseparable $\epsilon < \epsilon_0$.

SVR constructs a regression model that minimizes some empirical risk measure regularized to control capacity. Let x be the $n$ predictor variables and $y$ the dependent response variable. In [6], Vapnik proposed using the $\epsilon$-insensitive loss function $L^\epsilon(\mathrm{x}, y, f) = |y - f(\mathrm{x})|_\epsilon = \max(0, |y - f(\mathrm{x})| - \epsilon)$, in which an example is in error if its residual $|y - f(\mathrm{x})|$ is greater than $\epsilon$. Plotting the points in $(\mathrm{x}, y)$ space as in Figure 1(a), we see that for a "perfect" regression model the data fall in a hard $\epsilon$-tube about the regression line. Let $(X_i, y_i)$ be an example where $i = 1, 2, \cdots, m$, $X_i$ is the $i^{th}$ predictor vector, and $y_i$ is its response. The training data are then $(\mathbf{X}, \mathrm{y})$ where $X_i$ is a row of the matrix $\mathbf{X} \in R^{m \times n}$ and $\mathrm{y} \in R^m$ is the response. A hard $\epsilon$-tube for a fixed $\epsilon > 0$ is defined as a plane $y = \mathrm{w}'\mathrm{x} + b$ satisfying $-\epsilon e \leq \mathrm{y} - \mathbf{X}\mathrm{w} - be \leq \epsilon e$ where $e$ is an $m$-dimensional vector of ones.

When does a hard $\epsilon$-tube exist? Clearly, for $\epsilon$ large enough such a tube always

exists for finite data. The smallest tube, the $\epsilon_0$-tube, can be found by optimizing:

$$\min_{w,b,\epsilon} \epsilon \quad s.t. -\epsilon e \leq y - \mathbf{X}w - be \leq \epsilon e \quad \quad (1)$$

Note that the smallest tube is typically not the $\epsilon$-SVR solution. Let $\mathcal{D}^+$ and $\mathcal{D}^-$ be formed by augmenting the data with the response variable respectively increased and decreased by $\epsilon$, i.e. $\mathcal{D}^+ = \{(X_i, y_i + \epsilon),\ i = 1, \cdots, m\}$ and $\mathcal{D}^- = \{(X_i, y_i - \epsilon),\ i = 1, \cdots, m\}$. Consider the simple problem in Figure 1(a). For any fixed $\epsilon > 0$, there are three possible cases: $\epsilon > \epsilon_0$ in which strict hard $\epsilon$-tubes exist, $\epsilon = \epsilon_0$ in which only $\epsilon_0$-tubes exist, and $\epsilon < \epsilon_0$ in which no hard $\epsilon$-tubes exist. A strict hard $\epsilon$-tube with no points on the edges of the tube only exists for $\epsilon > \epsilon_0$. Figure 1(b-d) illustrates what happens in the dual space for each case. The convex hulls of $\mathcal{D}^+$ and $\mathcal{D}^-$ are drawn along with the max-margin plane in (b) and the supporting plane in (c) for separating the convex hulls.

Clearly, the existence of the tube is directly related to the separability of $\mathcal{D}^+$ and $\mathcal{D}^-$. If $\epsilon > \epsilon_0$ then a strict tube exists and the convex hulls of $\mathcal{D}^+$ and $\mathcal{D}^-$ are strictly separable[1]. There are infinitely many possible $\epsilon$-tubes when $\epsilon > \epsilon_0$. One can see that the max-margin plane separating $\mathcal{D}^+$ and $\mathcal{D}^-$ corresponds to one such $\epsilon$. In fact this plane forms an $\hat{\epsilon}$ tube where $\epsilon > \hat{\epsilon} \geq \epsilon_0$. If $\epsilon = \epsilon_0$, then the convex hulls of $\mathcal{D}^+$ and $\mathcal{D}^-$ are separable but not strictly separable. The plane that separates the two convex hulls forms the $\epsilon_0$ tube. In the last case, where $\epsilon < \epsilon_0$, the two sets $\mathcal{D}^+$ and $\mathcal{D}^-$ intersect. No $\epsilon$-tubes or max-margin planes exist.

It is easy to show by construction that if a hard $\epsilon$-tube exists for a given $\epsilon > 0$ then the convex hulls of $\mathcal{D}^+$ and $\mathcal{D}^-$ will be separable. If a hard $\epsilon$-tube exists, then there exists $(w, b)$ such that

$$(y + \epsilon e) - \mathbf{X}w - be \geq 0, \quad (y - \epsilon e) - \mathbf{X}w - be \leq 0. \quad \quad (2)$$

For any convex combination of $\mathcal{D}^+$, $\left(\begin{smallmatrix}\mathbf{X}' \\ (y+\epsilon e)'\end{smallmatrix}\right)u$ where $e'u = 1$, $u \geq 0$ of points $(X_i, y_i + \epsilon)$, $i = 1, 2, \cdots, m$, we have $(y + \epsilon e)'u - w'(\mathbf{X}'u) - b \geq 0$. Similarly for $\mathcal{D}^-$, $\left(\begin{smallmatrix}\mathbf{X}' \\ (y-\epsilon e)'\end{smallmatrix}\right)v$ where $e'v = 1$, $v \geq 0$ of points $(X_i, y_i - \epsilon)$, $i = 1, 2, \cdots, m$, we have $(y - \epsilon e)'v - w'(\mathbf{X}'v) - b \leq 0$. Then the plane $y = w'x + b$ in the $\epsilon$-tube separates the two convex hulls. Note the separating plane and the $\epsilon$-tube plane are the same. If no separating plane exists, then there is no tube. Gale's Theorem[2] of the alternative can be used to precisely characterize the $\epsilon$-tube.

**Theorem 2.1 (Conditions for existence of hard $\epsilon$-tube)** *A hard $\epsilon$-tube exists for a given $\epsilon > 0$ if and only if the following system in $(u, v)$ has no solution:*

$$\mathbf{X}'u = \mathbf{X}'v, \quad e'u = e'v = 1, \quad (y + \epsilon e)'u - (y - \epsilon e)'v < 0, \quad u \geq 0, v \geq 0. \quad (3)$$

**Proof** A hard $\epsilon$-tube exists if and only if System (2) has a solution. By Gale's Theorem of the alternative [4], system (2) has a solution if and only if the following alternative system has no solution: $\mathbf{X}'u = \mathbf{X}'v$, $e'u = e'v$, $(y + \epsilon e)'u - (y - \epsilon e)'v = -1$, $u \geq 0, v \geq 0$. Rescaling by $\frac{1}{\sigma}$ where $\sigma = e'u = e'v > 0$ yields the result. ∎

Note that if $\epsilon \geq \epsilon_0$ then $(y + \epsilon e)'u - (y - \epsilon e)'v \geq 0$. for any $(u, v)$ such that $\mathbf{X}'u = \mathbf{X}'v$, $e'u = e'v = 1$, $u, v \geq 0$. So as a consequence of this theorem, if $\mathcal{D}^+$ and $\mathcal{D}^-$ are separable, then a hard $\epsilon$-tube exists.

## 3   Constructing the $\epsilon$-tube

For any $\epsilon > \epsilon_0$ infinitely many possible $\epsilon$-tubes exist. Which $\epsilon$-tube should be used? The linear program (1) can be solved to find the smallest $\epsilon_0$-tube. But this corresponds to just doing empirical risk minimization and may result in poor generalization due to overfitting. We know capacity control or structural risk minimization is fundamental to the success of SVM classification and regression.

We take our inspiration from SVM classification. In hard-margin SVM classification, the dual SVM formulation constructs the max-margin plane by finding the two nearest points in the convex hulls of the two classes. The max-margin plane is the plane bisecting these two points. We know that the existence of the tube is linked to the separability of the shifted sets, $\mathcal{D}^+$ and $\mathcal{D}^-$. The key insight is that the regression problem can be regarded as a classification problem between $\mathcal{D}^+$ and $\mathcal{D}^-$. The two sets $\mathcal{D}^+$ and $\mathcal{D}^-$ defined as in Section 2 both contain the same number of data points. The only significant difference occurs along the $y$ dimension as the response variable $y$ is shifted up by $\epsilon$ in $\mathcal{D}^+$ and down by $\epsilon$ in $\mathcal{D}^-$. For $\epsilon > \epsilon_0$, the max-margin separating plane corresponds to a hard $\hat{\epsilon}$-tube where $\epsilon > \hat{\epsilon} \geq \epsilon_0$. The resulting tube is smaller than $\epsilon$ but not necessarily the smallest tube. Figure 1(b) shows the max-margin plane found for $\epsilon > \epsilon_0$. Figure 1(a) shows that the corresponding linear regression function for this simple example turns out to be the $\epsilon_0$ tube. As in classification, we will have a hard and soft $\epsilon$-tube case. The soft $\epsilon$-tube with $\epsilon \leq \epsilon_0$ is used to obtain good generalization when there are outliers.

### 3.1   The hard $\epsilon$-tube case

We now apply the dual convex hull method to constructing the max-margin plane for our augmented sets $\mathcal{D}^+$ and $\mathcal{D}^-$ assuming they are strictly separable, i.e. $\epsilon > \epsilon_0$. The problem is illustrated in detail in Figure 2. The closest points of $\mathcal{D}^+$ and $\mathcal{D}^-$ can be found by solving the following dual C-SVR quadratic program:

$$\min_{u,v} \quad \tfrac{1}{2} \left\| \left( _{(y+\epsilon e)'}^{\mathbf{X}'} \right) u - \left( _{(y-\epsilon e)'}^{\mathbf{X}'} \right) v \right\|^2 \tag{4}$$
$$s.t. \quad e'u = 1, \ e'v = 1, \ u \geq 0, \ v \geq 0.$$

Let the closest points in the convex hulls of $\mathcal{D}^+$ and $\mathcal{D}^-$ be $c = \left( _{(y+\epsilon e)'}^{\mathbf{X}'} \right) \hat{u}$ and $d = \left( _{(y-\epsilon e)'}^{\mathbf{X}'} \right) \hat{v}$ respectively. The max-margin separating plane bisects these two points. The normal $(\hat{w}, \hat{\delta})$ of the plane is the difference between them, i.e., $\hat{w} = \mathbf{X}'\hat{u} - \mathbf{X}'\hat{v}$, $\hat{\delta} = (y + \epsilon e)'\hat{u} - (y - \epsilon e)'\hat{v}$. The threshold, $\hat{b}$, is the distance from the origin to the point halfway between the two closest points along the normal: $\hat{b} = \hat{w}' \left( \frac{\mathbf{X}'\hat{u} + \mathbf{X}'\hat{v}}{2} \right) + \hat{\delta} \left( \frac{y'\hat{u} + y'\hat{v}}{2} \right)$. The separating plane has the equation $\hat{w}'x + \hat{\delta}y - \hat{b} = 0$. Rescaling this plane yields the regression function.

Dual C-SVR (4) is in the dual space. The corresponding Primal C-SVR is:

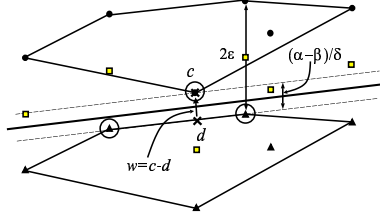

Figure 2: The solution $\hat{\epsilon}$-tube found by C-SVR can have $\hat{\epsilon} < \epsilon$. Squares are original data. Dots are in $\mathcal{D}^+$. Triangles are in $\mathcal{D}^-$ . Support Vectors are circled.

$$\min_{w,\delta,\alpha,\beta} \quad \tfrac{1}{2}\|\mathbf{w}\|^2 + \tfrac{1}{2}\delta^2 - (\alpha - \beta)$$
$$s.t. \quad \begin{aligned} \mathbf{X}w + \delta(y + \epsilon e) - \alpha e \geq 0 \\ \mathbf{X}w + \delta(y - \epsilon e) - \beta e \leq 0. \end{aligned} \tag{5}$$

Dual C-SVR (4) can be derived by taking the Wolfe or Lagrangian dual [4] of primal C-SVR (5) and simplifying.

We prove that the optimal plane from C-SVR bisects the $\hat{\epsilon}$ tube. The supporting planes for class $\mathcal{D}^+$ and class $\mathcal{D}^-$ determines the lower and upper edges of the $\hat{\epsilon}$-tube respectively. The support vectors from $\mathcal{D}^+$ and $\mathcal{D}^-$ correspond to the points along the lower and upper edges of the $\hat{\epsilon}$-tube. See Figure 2.

**Theorem 3.1 (C-SVR constructs $\hat{\epsilon}$-tube)** *Let the max-margin plane obtained by C-SVR (4) be $\hat{w}'x + \hat{\delta}y - \hat{b} = 0$ where $\hat{w} = \mathbf{X}'\hat{u} - \mathbf{X}'\hat{v}$, $\hat{\delta} = (y + \epsilon e)'\hat{u} - (y - \epsilon e)'\hat{v}$, and $\hat{b} = \hat{w}'\left(\frac{\mathbf{X}'\hat{u} + \mathbf{X}'\hat{v}}{2}\right) + \hat{\delta}\left(\frac{y'\hat{u} + y'\hat{v}}{2}\right)$. If $\epsilon > \epsilon_0$, then the plane $y = w'x + b$ corresponds to an $\hat{\epsilon}$-tube of training data $(X_i, y_i)$, $i = 1, 2, \cdots, m$ where $w = -\frac{\hat{w}}{\hat{\delta}}$, $b = \frac{\hat{b}}{\hat{\delta}}$ and $\hat{\epsilon} = \epsilon - \frac{\hat{\alpha} - \hat{\beta}}{2\hat{\delta}} < \epsilon$.*

**Proof** First, we show $\hat{\delta} > 0$. By the Wolfe duality theorem [4], $\hat{\alpha} - \hat{\beta} > 0$, since the objective values of (5) and the negative objective value of (4) are equal at optimality. By complementarity, the closest points are right on the margin planes $\hat{w}'x + \hat{\delta}y - \hat{\alpha} = 0$ and $\hat{w}'x + \hat{\delta}y - \hat{\beta} = 0$ respectively, so $\hat{\alpha} = \hat{w}'\mathbf{X}'\hat{u} + \hat{\delta}(y + \epsilon e)'\hat{u}$ and $\hat{\beta} = \hat{w}'\mathbf{X}'\hat{v} + \hat{\delta}(y - \epsilon e)'\hat{v}$. Hence $\hat{b} = \frac{\hat{\alpha} + \hat{\beta}}{2}$, and $\hat{w}$, $\hat{\delta}$, $\hat{\alpha}$, and $\hat{\beta}$ satisfy the constraints of problem (5), i.e., $\mathbf{X}\hat{w} + \hat{\delta}(y + \epsilon e) - \hat{\alpha}e \geq 0$, $\mathbf{X}\hat{w} + \hat{\delta}(y - \epsilon e) - \hat{\beta}e \leq 0$. Then subtract the second inequality from the first inequality: $2\hat{\delta}\epsilon - \hat{\alpha} + \hat{\beta} \geq 0$, that is, $\hat{\delta} \geq \frac{\hat{\alpha} - \hat{\beta}}{2\epsilon} > 0$ because $\epsilon > \epsilon_0 \geq 0$. Rescale constraints by $-\hat{\delta} < 0$, and reverse the signs. Let $w = -\frac{\hat{w}}{\hat{\delta}}$, then the inequalities become $\mathbf{X}w - y \leq \epsilon e - \frac{\hat{\alpha}}{\hat{\delta}}e$, $\mathbf{X}w - y \geq -\epsilon e - \frac{\hat{\beta}}{\hat{\delta}}e$. Let $b = \frac{\hat{b}}{\hat{\delta}}$, then $\frac{\hat{\alpha}}{\hat{\delta}} = b + \frac{\hat{\alpha} - \hat{\beta}}{2\hat{\delta}}$ and $\frac{\hat{\beta}}{\hat{\delta}} = b - \frac{\hat{\alpha} - \hat{\beta}}{2\hat{\delta}}$. Substituting into the previous inequalities yields $\mathbf{X}w - y \leq \left(\epsilon - \frac{\hat{\alpha} - \hat{\beta}}{2\hat{\delta}}\right)e - be$, $\mathbf{X}w - y \geq -\left(\epsilon - \frac{\hat{\alpha} - \hat{\beta}}{2\hat{\delta}}\right)e - be$. Denote $\hat{\epsilon} = \epsilon - \frac{\hat{\alpha} - \hat{\beta}}{2\hat{\delta}} < \epsilon$. These inequalities become $\mathbf{X}w + be - y \leq \hat{\epsilon}e$, $\mathbf{X}w + be - y \geq -\hat{\epsilon}e$. Hence the plane $y = w'x + b$ is in the middle of the $\hat{\epsilon} < \epsilon$ tube. ∎

## 3.2   The soft $\epsilon$-tube case

For $\epsilon < \epsilon_0$, a hard $\epsilon$-tube does not exist. Making $\epsilon$ large to fit outliers may result in poor overall accuracy. In soft-margin classification, outliers were handled in the

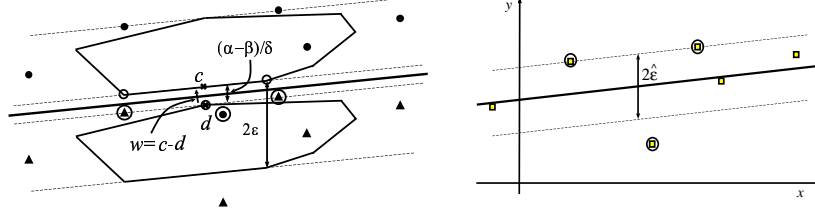

Figure 3: Soft $\hat{\epsilon}$-tube found by RC-SVR: *left*: dual, *right*: primal space.

dual space by using reduced convex hulls. The same strategy works for soft $\epsilon$-tubes, see Figure 3. Instead of taking the full convex hulls of $\mathcal{D}^+$ and $\mathcal{D}^-$ , we reduce the convex hulls away from the difficult boundary cases. RC-SVR computes the closest points in the reduced convex hulls

$$\min_{\mathbf{u},\mathbf{v}} \quad \tfrac{1}{2}\left\|\left(\genfrac{}{}{0pt}{}{\mathbf{X}'}{(\mathbf{y}+\epsilon\mathbf{e})'}\right)\mathbf{u} - \left(\genfrac{}{}{0pt}{}{\mathbf{X}'}{(\mathbf{y}-\epsilon\mathbf{e})'}\right)\mathbf{v}\right\|^2 \tag{6}$$
$$s.t. \quad \mathbf{e}'\mathbf{u} = 1, \ \mathbf{e}'\mathbf{v} = 1, \ 0 \le \mathbf{u} \le D\mathbf{e}, \ 0 \le \mathbf{v} \le D\mathbf{e}.$$

Parameter $D$ determines the robustness of the solution by reducing the convex hull. $D$ limits the influence of any single point. As in $\nu$-SVM, we can parameterize $D$ by $\nu$. Let $D = \frac{1}{\nu m}$ where $m$ is the number of points. Figure 3 illustrates the case for $m = 6$ points, $\nu = 2/6$, and $D = 1/2$. In this example, every point in the reduced convex hull must depend on at least two data points since $\sum_{i=1}^{m} u_i = 1$ and $0 \le u_i \le 1/2$. In general, every point in the reduced convex hull can be written as the convex combination of at least $\lceil 1/D \rceil = \lceil \nu * m \rceil$. Since these points are exactly the support vectors and there are two reduced convex hulls, $2 * \lceil \nu m \rceil$ is a lower bound on the number of support vectors in RC-SVR. By choosing $\nu$ sufficiently large, the inseparable case with $\epsilon \le \epsilon_0$ is transformed into a separable case where once again our nearest-points-in-the-convex-hull-problem is well defined.

As in classification, the dual reduced convex hull problem corresponds to computing a soft $\epsilon$-tube in the primal space. Consider the following soft tube version of the primal C-SVR (7) which has its Wolfe Dual RC-SVR (6):

$$\min_{\mathbf{w},\delta,\alpha,\beta,\boldsymbol{\xi},\boldsymbol{\eta}} \quad \tfrac{1}{2}\|\mathbf{w}\|^2 + \tfrac{1}{2}\delta^2 - (\alpha - \beta) + C(\mathbf{e}'\boldsymbol{\xi} + \mathbf{e}'\boldsymbol{\eta})$$
$$s.t. \quad \mathbf{X}\mathbf{w} + \delta(\mathbf{y} + \epsilon\mathbf{e}) - \alpha\mathbf{e} + \boldsymbol{\xi} \ge 0, \ \boldsymbol{\xi} \ge 0 \tag{7}$$
$$\mathbf{X}\mathbf{w} + \delta(\mathbf{y} - \epsilon\mathbf{e}) - \beta\mathbf{e} - \boldsymbol{\eta} \le 0, \ \boldsymbol{\eta} \ge 0$$

The results of Theorem 3.1 can be easily extended to soft $\epsilon$-tubes.

**Theorem 3.2 (RC-SVR constructs soft $\hat{\epsilon}$-tube)** *Let the soft max-margin plane obtained by RC-SVR (6) be* $\hat{\mathbf{w}}'\mathbf{x} + \hat{\delta}y - \hat{b} = 0$ *where* $\hat{\mathbf{w}} = \mathbf{X}'\hat{\mathbf{u}} - \mathbf{X}'\hat{\mathbf{v}}$, $\hat{\delta} = (\mathbf{y} + \epsilon\mathbf{e})'\hat{\mathbf{u}} - (\mathbf{y} - \epsilon\mathbf{e})'\hat{\mathbf{v}}$, *and* $\hat{b} = \left(\frac{\mathbf{X}'\hat{\mathbf{u}} + \mathbf{X}'\hat{\mathbf{v}}}{2}\right)'\hat{\mathbf{w}} + \left(\frac{\mathbf{y}'\hat{\mathbf{u}} + \mathbf{y}'\hat{\mathbf{v}}}{2}\right)\hat{\delta}$. *If* $0 < \epsilon \le \epsilon_0$, *then the plane* $y = \mathbf{w}'\mathbf{x} + b$ *corresponds to a soft* $\hat{\epsilon} = \epsilon - \frac{\tilde{\alpha} - \tilde{\beta}}{2\hat{\delta}} < \epsilon$*-tube of training data* $(X_i, y_i)$, $i = 1, 2, \cdots, m$, *i.e., a* $\hat{\epsilon}$*-tube of reduced convex hull of training data where* $\mathbf{w} = -\frac{\hat{\mathbf{w}}}{\hat{\delta}}$, $b = \frac{\hat{b}}{\hat{\delta}}$ *and* $\tilde{\alpha} = \hat{\mathbf{w}}'\mathbf{X}'\hat{\mathbf{u}} + \hat{\delta}(\mathbf{y} + \epsilon\mathbf{e})'\hat{\mathbf{u}}$, $\tilde{\beta} = \hat{\mathbf{w}}'\mathbf{X}'\hat{\mathbf{v}} + \hat{\delta}(\mathbf{y} - \epsilon\mathbf{e})'\hat{\mathbf{v}}$.

Notice that the $\tilde{\alpha}$ and $\tilde{\beta}$ determine the planes parallel to the regression plane and through the closest points in each reduced convex hull of shifted data. In the

inseparable case, these planes are parallel but not necessarily identical to the planes obtained by the primal RC-SVR (7).

Nonlinear C-SVR and RC-SVR can be achieved by using the usual kernel trick. Let $\Phi$ by a nonlinear mapping of x such that $k(X_i, X_j) = \Phi(X_i) \cdot \Phi(X_j)$. The objective function of C-SVR (4) and RC-SVR (6) applied to the mapped data becomes

$$
\begin{aligned}
&\tfrac{1}{2} \sum_{i=1}^{m} \sum_{j=1}^{m} \left((u_i - v_i)(u_j - v_j)(\Phi(X_i) \cdot \Phi(X_j) + y_i y_j)\right) + 2\epsilon \sum_{i=1}^{m} \left(y_i(u_i - v_i)\right) \\
&= \tfrac{1}{2} \sum_{i=1}^{m} \sum_{j=1}^{m} \left((u_i - v_i)(u_j - v_j)(k(X_i, X_j) + y_i y_j)\right) + 2\epsilon \sum_{i=1}^{m} \left(y_i(u_i - v_i)\right)
\end{aligned} \tag{8}
$$

The final regression model after optimizing C-SVR or RC-SVR with kernels takes the form of $f(\mathrm{x}) = \sum_{i=1}^{m} (\bar{u}_i - \bar{v}_i) k(X_i, \mathrm{x}) + \bar{b}$, where $\bar{u}_i = \frac{\hat{u}_i}{\hat{\delta}}$, $\bar{v}_i = \frac{\hat{v}_i}{\hat{\delta}}$, $\hat{\delta} = (\hat{\mathrm{u}} - \hat{\mathrm{v}})'\mathrm{y} + 2\epsilon$, and the intercept term $\bar{b} = \frac{(\hat{\mathrm{u}}+\hat{\mathrm{v}})'\mathbf{K}(\hat{\mathrm{u}}-\hat{\mathrm{v}})}{2\hat{\delta}} + \frac{(\hat{\mathrm{u}}+\hat{\mathrm{v}})'\mathrm{y}}{2}$ where $\mathbf{K}_{ij} = k(X_i, X_j)$.

## 4  Computational Results

We illustrate the difference between RC-SVR and $\epsilon$-SVR on a toy linear problem[3]. Figure 4 depicts the functions constructed by RC-SVR and $\epsilon$-SVR for different values of $\epsilon$. For large $\epsilon$, $\epsilon$-SVR produces undesirable results. RC-SVR constructs the same function for $\epsilon$ sufficiently large. Too small $\epsilon$ can result in poor generalization.

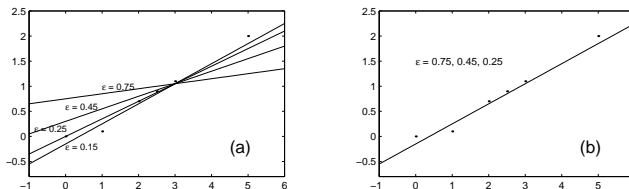

Figure 4: Regression lines from (a) $\epsilon$-SVR and (b) RC-SVR with distinct $\epsilon$.

In Table 1, we compare RC-SVR, $\epsilon$-SVR and $\nu$-SVR on the Boston Housing problem. Following the experimental design in [5] we used RBF kernel with $2\sigma^2 = 3.9$, $C = 500 \cdot m$ for $\epsilon$-SVR and $\nu$-SVR, and $\epsilon = 3.0$ for RC-SVR. RC-SVR, $\epsilon$-SVR, and $\nu$-SVR are computationally similar for good parameter choices. In $\epsilon$-SVR, $\epsilon$ is fixed. In RC-SVR, $\epsilon$ is the maximum allowable tube width. Choosing $\epsilon$ is critical for $\epsilon$-SVR but less so for RC-SVR. Both RC-SVR and $\nu$-SVR can shrink or grow the tube according to desired robustness. But $\nu$-SVR has no upper $\epsilon$ bound.

## 5  Conclusion and Discussion

By examining when $\epsilon$-tubes exist, we showed that in the dual space SVR can be regarded as a classification problem. Hard and soft $\epsilon$-tubes are constructed by separating the convex or reduced convex hulls respectively of the training data with the response variable shifted up and down by $\epsilon$. We proposed RC-SVR based on choosing the soft max-margin plane between the two shifted datasets. Like $\nu$-SVM, RC-SVR shrinks the $\epsilon$-tube. The max-margin determines how much the tube can shrink. Domain knowledge can be incorporated into the RC-SVR parameters $\epsilon$

Table 1: Testing Results for Boston Housing, MSE= average of mean squared errors of 25 testing points over 100 trials, STD: standard deviation

| | $2\nu$ | 0.1 | 0.2 | 0.3 | 0.4 | 0.5 | 0.6 | 0.7 | 0.8 |
|---|---|---|---|---|---|---|---|---|---|
| RC-SVR | MSE | 37.3 | 11.2 | 10.7 | 9.6 | 8.9 | 10.6 | 11.5 | 12.5 |
| | STD | 72.3 | 7.6 | 7.3 | 7.4 | 8.4 | 9.1 | 9.3 | 9.8 |
| | $\epsilon$ | 0 | 1 | 2 | 3 | 4 | 5 | 6 | 7 |
| $\epsilon$-SVR | MSE | 11.2 | 10.8 | 9.5 | 10.3 | 11.6 | 13.6 | 15.6 | 17.2 |
| | STD | 8.3 | 8.2 | 8.2 | 7.3 | 5.8 | 5.8 | 5.9 | 5.8 |
| | $\nu$ | 0.1 | 0.2 | 0.3 | 0.4 | 0.5 | 0.6 | 0.7 | 0.8 |
| $\nu$-SVR | MSE | 9.6 | 8.9 | 9.5 | 10.8 | 10.9 | 11.0 | 11.2 | 11.1 |
| | STD | 5.8 | 7.9 | 8.3 | 8.2 | 8.3 | 8.4 | 8.5 | 8.4 |

and $\nu$. The parameter $C$ in $\nu$-SVM and $\epsilon$-SVR has been eliminated. Computationally, no one method is superior for good parameter choices. RC-SVR alone has a geometrically intuitive framework that allows users to easily grasp the model and its parameters. Also, RC-SVR can be solved by fast nearest point algorithms. Considering regression as a classification problem suggests other interesting SVR formulations. We can show $\epsilon$-SVR is equivalent to finding closest points in a reduced convex hull problem for certain $C$, but the equivalent problem utilizes a different metric in the objective function than RC-SVR. Perhaps other variations would yield even better formulations.

**Acknowledgments**

Thanks to referees and Bernhard Schölkopf for suggestions to improve this work. This work was supported by NSF IRI-9702306, NSF IIS-9979860.

## Footnotes

[1] We use the following definitions of separation of convex sets. Let $\mathcal{D}^+$ and $\mathcal{D}^-$ be nonempty convex sets. A plane $H = \{x : w'x = \alpha\}$ is said to separate $\mathcal{D}^+$ and $\mathcal{D}^-$ if $w'x \geq \alpha, \forall x \in \mathcal{D}^+$ and $w'x \leq \alpha, \forall x \in \mathcal{D}^-$. $H$ is said to strictly separate $\mathcal{D}^+$ and $\mathcal{D}^-$ if $w'x \geq \alpha + \Delta$ for $x \in \mathcal{D}^+$, and $w'x \leq \alpha - \Delta$ for each $x \in \mathcal{D}^-$ where $\Delta$ is a positive scalar.

[2] The system $\mathbf{A}x \leq c$ has a (or has no) solution if and only if the alternative system $\mathbf{A}'y = 0$, $c'y = -1$, $y \geq 0$ has no (or has a) solution.

[3]The data consist of $(x, y)$: (0 0), (1 0.1), (2 0.7), (2.5 0.9), (3 1.1) and (5 2). The CPLEX 6.6 optimization package was used.

# References

[1] K. Bennett and E. Bredensteiner. Duality and Geometry in SVM Classifiers. In P. Langley, eds., *Proc. of Seventeenth Intl. Conf. on Machine Learning*, p 57–64, Morgan Kaufmann, San Francisco, 2000.

[2] D. Crisp and C. Burges. A Geometric Interpretation of $\nu$-SVM Classifiers. In S. Solla, T. Leen, and K. Muller, eds., *Advances in Neural Info. Proc. Sys.*, Vol 12. p 244–251, MIT Press, Cambridge, MA, 1999.

[3] S.S. Keerthi, S.K. Shevade, C. Bhattacharyya and K.R.K. Murthy, A Fast Iterative Nearest Point Algorithm for Support Vector Machine Classifier Design, IEEE Transactions on Neural Networks, Vol. 11, pp.124-136, 2000.

[4] O. Mangasarian. *Nonlinear Programming.* SIAM, Philadelphia, 1994.

[5] B. Schölkopf, P. Bartlett, A. Smola and R. Williamson. Shrinking the Tube: A New Support Vector Regression Algorithm. In M. Kearns, S. Solla, and D. Cohn eds., *Advances in Neural Info. Proc. Sys.*, Vol 12, MIT Press, Cambridge, MA, 1999.

[6] V. Vapnik. *The Nature of Statistical Learning Theory.* Wiley, New York, 1995.